# Interference in Learning Internal Models of Inverse Dynamics in Humans

Reza Shadmehr,* Tom Brashers-Krug, and Ferdinando Mussa-Ivaldi[†]
Dept. of Brain and Cognitive Sciences
M. I. T., Cambridge, MA 02139
reza@bme.jhu.edu, tbk@ai.mit.edu, sandro@parker.physio.nwu.edu

## Abstract

Experiments were performed to reveal some of the computational properties of the human motor memory system. We show that as humans practice reaching movements while interacting with a novel mechanical environment, they learn an *internal model* of the inverse dynamics of that environment. Subjects show recall of this model at testing sessions 24 hours after the initial practice. The representation of the internal model in memory is such that there is interference when there is an attempt to learn a new inverse dynamics map immediately after an anticorrelated mapping was learned. We suggest that this interference is an indication that the same computational elements used to encode the first inverse dynamics map are being used to learn the second mapping. We predict that this leads to a forgetting of the initially learned skill.

## 1 Introduction

In tasks where we use our hands to interact with a tool, our motor system develops a model of the dynamics of that tool and uses this model to control the coupled dynamics of our arm and the tool (Shadmehr and Mussa-Ivaldi 1994). In physical systems theory, the tool is a mechanical analogue of an admittance, mapping a force as input onto a change in state as output (Hogan 1985). In this framework, the

[†]Currently at Dept. Physiology, Northwestern Univ Med Sch (M211), Chicago, IL 60611

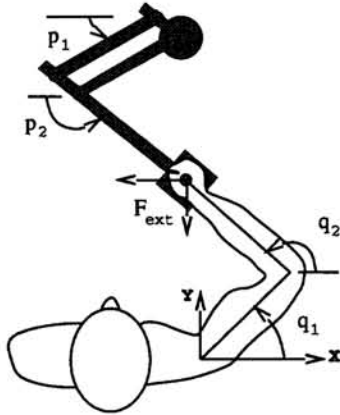

**Figure 1:** The experimental setup. The robot is a very low friction planar mechanism powered by two torque motors that act on the shoulder and elbow joints. Subject grips the end-point of the robot which houses a force transducer and moves the hand to a series of targets displayed on a monitor facing the subject (not shown). The function of the robot is to produce novel force fields that the subject learns to compensate for during reaching movements.

model developed by the motor control system during the learning process needs to approximate an inverse of this mapping. This inverse dynamics map is called an *internal model* of the tool.

We have been interested in understanding the representations that the nervous system uses in learning and storing such internal models. In a previous work we measured the way a learned internal model extrapolated beyond the training data (Shadmehr and Mussa-Ivaldi 1994). The results suggested that the coordinate system of the learned map was in intrinsic (e.g., joint or muscles based) rather than in extrinsic (e.g., hand based) coordinates. Here we present a mathematical technique to estimate the input–output properties of the learned map. We then explore the issue of how the motor memory might store two maps which have similar inputs but different outputs.

## 2   Quantifying the internal model

In our paradigm, subjects learn to control an artificial tool: the tool is a robot manipulandum which has torque motors that can be programmed to produce a variety of dynamical environments (Fig. 1). The task for the subject is to grasp the end-effector and make point to point reaching movements to a series of targets. The environments are represented as force fields acting on the subject's hand, and a typical case is shown in Fig. 2A. A typical experiment begins with the robot motors turned off. In this "null" environment subjects move their hand to the targets in a smooth, straight line fashion. When the force field is introduced, the dynamics of the task change and the hand trajectory is significantly altered (Shadmehr and Mussa-Ivaldi 1994). With practice (typically hundreds of movements), hand trajectories return to their straight line path. We have suggested that practice leads to formation of an internal model which functions as an inverse dynamics mapping, i.e., from a desired trajectory (presumably in terms of hand position and velocity, Wolpert et al. 1995) to a prediction of forces that will be encountered along the trajectory. We designed a method to quantify these forces and estimate the output properties of the internal model.

If we position a force transducer at the interaction point between the robot and the subject, we can write the dynamics of the four link system in Fig. 1 in terms of the

following coupled vector differential equations:

$$I_r(p)\ddot{p} + G_r(p,\dot{p})\dot{p} = E(p,\dot{p}) + J_r^T F \tag{1}$$

$$I_s(q)\ddot{q} + G_s(q,\dot{q})\dot{q} = C(q,\dot{q},q^*(t)) - J_s^T F \tag{2}$$

where $I$ and $G$ are inertial and Corriolis/centripetal matrix functions, $E$ is the torque field produced by the robot's motors, i.e., the environment, $F$ is the force measured at the handle of the robot, $C$ is the controller implemented by the motor system of the subject, $q^*(t)$ is the reference trajectory planned by the motor system of the subject, $J$ is the Jacobian matrix describing the differential transformation of coordinates from endpoint to joints, $q$ and $p$ are joint positions of the subject and the robot, and the subscripts $s$ and $r$ denote subject or robot matrices.

In the null environment, i.e., $E = 0$ in Eq. (1), a solution to this coupled system is $q = q^*(t)$ and the arm follows the reference trajectory (typically a straight hand path with a Gaussian tangential velocity profile). Let us name the controller which accomplishes this task $C = C_0$ in Eq. (2). When the robot motors are producing a force field $E \neq 0$, it can be shown that the solution is $q = q^*(t)$ if and only if the new controller in Eq. (2) is $C = C_1 = C_0 + J_s^T J_r^{-T} E$. The internal model composed by the subject is $C_1 - C_0$, i.e., the change in the controller after some training period. We can estimate this quantity by measuring the change in the interaction force along a given trajectory before and after training. If we call these functions $F_0$ and $F_1$, then we have:

$$F_0(q,\dot{q},\ddot{q},q^*(t)) = J_s^{-T}(C_0 - I_s\ddot{q} - G_s\dot{q}) \tag{3}$$

$$F_1(q,\dot{q},\ddot{q},q^*(t)) = J_s^{-T}(C_0 + J_s^T J_r^{-T}\hat{E} - I_s\ddot{q} - G_s\dot{q}) \tag{4}$$

The functions $F_0$ and $F_1$ are impedances of the subject's arm as viewed from the interaction port. Therefore, by approximating the difference $F_1 - F_0$, we have an estimate of the change in the controller. The crucial assumption is that the reference trajectory $q^*(t)$ does not change during the training process.

In order to measure $F_0$, we had the subjects make movements in a series of environments. The environments were unpredictable (no opportunity to learn) and their purpose was to perturb the controller about the reference trajectory so we could measure $F_0$ at neighboring states. Next, the environment in Fig. 2A was presented and the subject given a practice period to adapt. After training, $F_1$ was estimated in a similar fashion as $F_0$. The difference between these two functions was calculated along all measured arm trajectories and the results were projected onto the hand velocity space. Due to computer limitations, only 9 trajectories for each target direction were used for this approximation. The resulting pattern of forces were interpolated via a sum of Gaussian radial basis functions, and are shown in Fig. 2B. This is the change in the impedance of the arm and estimates the input–output property of the internal model that was learned by this subject. We found that this subject, which provided some of best results in the test group, learned to change the effective impedance of his arm in a way that approximated the imposed force field. This would be a sufficient condition for the arm to compensate for the force field and allow the hand to follow the desired trajectory. An alternate strategy might have been to simply co-contract arm muscles: this would lead to an increased stiffness and an ability to resist arbitrary environmental forces. Figure 2B suggests that practice led to formation of an internal model specific to the dynamics of the imposed force field.

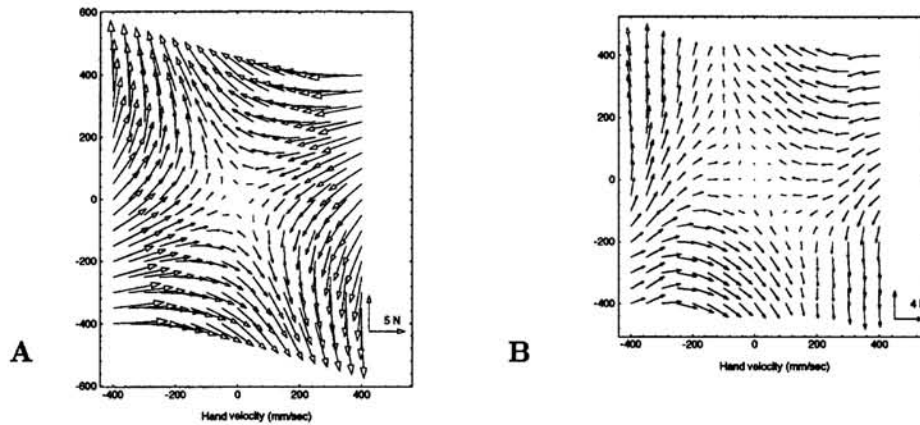

**Figure 2:** Quantification of the change in impedance of a subject's arm after learning a force field. **A:** The force field produced by the robot during the training period. **B:** The change in the subject's arm impedance after the training period, i.e., the internal model.

## 2.1  Formation of the internal model in long-term memory

Here we wished to determine whether subjects retained the internal model in long-term motor memory. We tested 16 naive subjects. They were instructed to move the handle of the robot to a sequence of targets in the null environment. Each movement was to last $500 \pm 50$ msec. They were given visual feedback on the timing of each movement. After 600 movements, subjects were able to consistently reach the targets in proper time. These trajectories constituted a baseline set.

Subjects returned the next day and were re-familiarized with the timing of the task. At this point a force field was introduced and subjects attempted to perform the exact task as before: get to the target in proper time. A sequence of 600 targets was given. When first introduced, the forces perturbed the subject's trajectories, causing them to deviate from the straight line path. As noted in previous work (Shadmehr and Mussa-Ivaldi 1994), these deviations decreased with practice. Eventually, subject's trajectories in the presence of the force field came to resemble those of the baseline, when no forces were present. The convergence of the trajectories to those performed at baseline is shown for all 16 subjects in Fig. 3A. The timing performance of the subjects while moving in the field is shown in Fig. 3B.

In order to determine whether subjects retained the internal model of the force field in long-term memory, we had them return the next day (24 to 30 hours later) and once again be tested on a force field. In half of the subjects, the force field presented was one that they had trained on in the previous day (call this field 1). In the other half, it was a force field which was novel to the subjects, field 2. Field 2 had a correlation value of $-1$ with respect to field 1 (i.e., each force vector in field 2 was a 180 degree rotation of the respective vector in field 1). Subjects who were tested on a field that they had trained on before performed significantly better ($p < 0.01$) than their initial performance (Fig. 4A), signifying retention. However, those who were given a field that was novel performed at naive levels (Fig. 4B). This result suggested that the internal model formed after practice in a given field was (1) specific to that field: performance on the untrained field was no better than

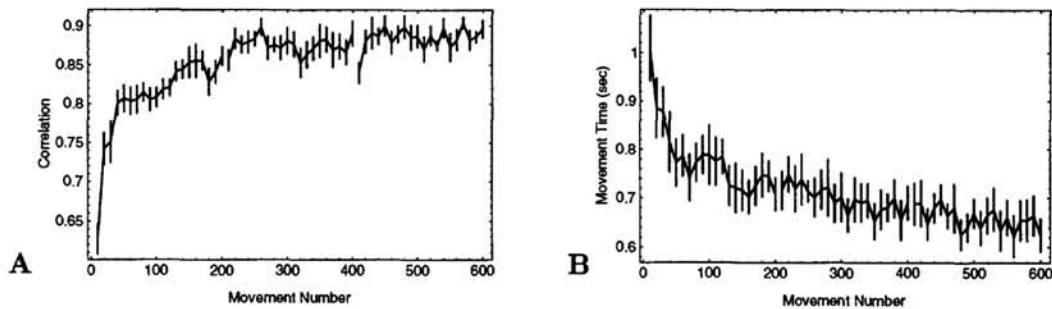

**Figure 3:** Measures of performance during the training period (600 movements) for 16 naive subjects. Short breaks (2 minutes) were given at intervals of 200 movements. **A:** Mean ± standard error (SE) of the correlation coefficient between hand trajectory in a null environment (called baseline trajectories, measured before exposure to the field), and trajectory in the force field. Hand trajectories in the field converge to that in the null field (i.e., become straight, with a bell shaped velocity profile). **B:** Mean ± SE of the movement period to reach a target. The goal was to reach the target in 0.5 ± 0.05 seconds.

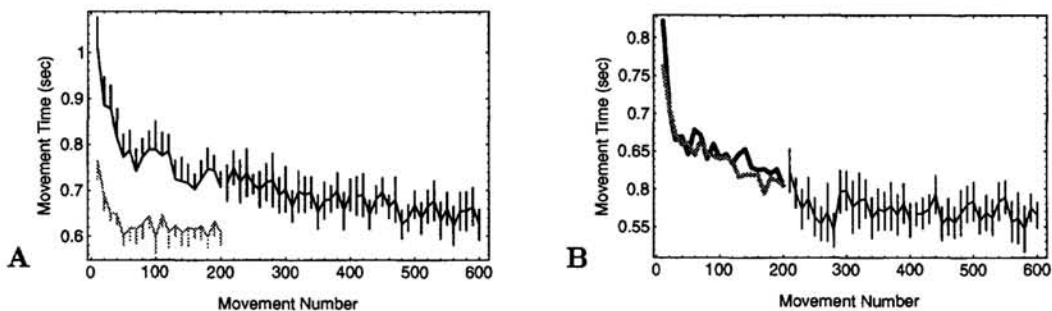

**Figure 4:** Subjects learned an internal model specific to the field and retained it in long-term memory. **A:** Mean ± standard error (SE) of the movement period in the force field (called field 1) during initial practice session (upper trace) and during a second session 24–30 hours after the initial practice (lower trace). **B:** Movement period in a different group of subjects during initial training (dark line) in field 1 and test in an anti-correlated field (called field 2) 24–30 hours later (gray line).

performance recorded in a separate set of naive subjects who were given than field in their initial training day; and (2) could be retained, as evidenced by performance in the following day.

## 2.2  Interference effects of the motor memory

In our experiment the "tool" that subjects learn to control is rather unusual, nevertheless, subjects learn its inverse dynamics and the memory is used to enhance performance 24 hours after its initial acquisition. We next asked how formation of this memory affected formation of subsequent internal models. In the previous section we showed that when a subject returns a day after the initial training, although the memory of the learned internal model is present, there is no interference (or decrement in performance) in learning a new, anti-correlated field. Here we show that when this temporal distance is significantly reduced, the just learned

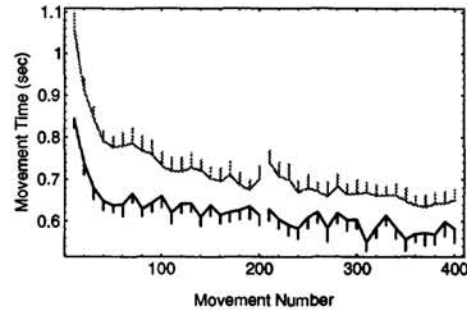

**Figure 5:** Interference in sequential learning of two uncorrelated force fields: The lower trace is the mean and standard error of the movement periods of a naive group of subjects during initial practice in a force field (called field 1). The upper trace is the movement period of another group of naive subjects in field 1, 5 minutes after practicing 400 movements in field 2, which was anti-correlated with field 1.

model interferes with learning of a new field.

Seven new subjects were recruited. They learned the timing of the task in a null environment and in the following day were given 400 targets in a force field (called field 1). They showed improvement in performance as before. After a short break (5–10 minutes in which they walked about the lab or read a magazine), they were given a new field: this field was called field 2 and was anti-correlated with respect to field 1. We found a significant reduction ($p < 0.01$) in their ability to learn field 2 (Fig. 5) when compared to a subject group which had not initially trained in field 1. In other words, performance in field 2 shortly after having learned field 1 was significantly worse than that of naives. Subjects seemed surprised by their inability to master the task in field 2. In order to demonstrate that field 2 *in isolation* was no more difficult to learn than field 1, we had a new set of subjects ($n = 5$) initially learn field 2, then field 1. Now we found a very large decrement in learnability of field 1.

One way to explain the decrement in performance shown in Fig. 5 is to assume that the same "computational elements" that represented the internal model of the first field were being used to learn the second field.[1] In other words, when the second field was given, because the forces were opposite to the first field, the internal model was badly biased against representing this second field: muscle torque patterns predicted for movement to a given target were in the wrong direction.

In the connectionist literature this is a phenomenon called *temporal interference* (Sutton 1986). As a network is trained, some of its elements acquire large weights and begin to dominate the input–output transformation. When a second task is presented with a new and conflicting map (mapping similar inputs to different outputs), there are large errors and the network performs more poorly than a "naive" network. As the network attempts to learn the new task, the errors are fed to each element (i.e., pre-synaptic input). This causes most activity in those elements that

had the largest synaptic weight. If the learning algorithm is Hebbian, i.e., weights change in proportion to co-activation of the pre- and the post-synaptic element, then the largest weights are changed the most, effectively causing a loss of what was learned in the first task. Therefore, from a computational stand point, we would expect that the internal model of field 1 as learned by our subjects should be destroyed by learning of field 2. Evidence for "catastrophic interference" in these subjects is presented elsewhere in this volume (Brashers-Krug et al. 1995).

The phenomenon of interference in sequential learning of two stimulus–response maps has been termed *proactive interference* or negative transfer in the psychological literature. In humans, interference has been observed extensively in verbal tasks involving short–term declarative memory (e.g., tasks involving recognition of words in a list or pairing of non-sense syllables, Bruce 1933, Melton and Irwin 1940, Sears and Hovland 1941). It has been found that interference is a function of the similarity of the stimulus–response maps in the two tasks: if the stimulus in the new learning task requires a response very different than what was recently learned, then there is significant interference. Interestingly, it has been shown that the amount of interference decreases with increased learning (or practice) on the first map (Siipola and Israel 1933).

In tasks involving procedural memory (which includes motor learning, Squire 1986), the question of interference has been controversial: Although Lewis et al. (1949) reported interference in sequential learning of two motor tasks which involved moving levers in response to a set of lights, it has been suggested that the interference that they observed might have been due to cognitive confusion (Schmidt 1988). In another study, Ross (1974) reported little interference in subjects learning her motor tasks.

We designed a task that had little or no cognitive components. We found that shortly after the acquisition of a motor memory, that memory strongly interfered with learning of a new, anti-correlated input–output mapping. However, this interference was not significant 24 hours after the memory was initially acquired. One possible explanation is that the initial learning has taken place in a temporary and vulnerable memory system. With time and/or practice, the information in this memory had transferred to long-term storage (Brashers-Krug et al. 1995).

Brain imaging studies during motor learning suggest that as subjects become more proficient in a motor task, neural fields in the motor cortex display increases in activity (Grafton et al. 1992) and new fields are recruited (Kawashima et al. 1994). It has been reported that when a subject attempts to learn two new motor tasks successively (in this case the tasks consisted of two sequences of finger movements), the neural activity in the motor cortex is lower for the second task, even when the order of the tasks is reversed (Jezzard et al. 1994). It remains to be seen whether this decrement in neural activity in the motor cortex is correlated with the interference observed when subjects attempt to learn two different input–output mappings in succession (Gandolfo et al. 1994).

## Footnotes

*Currently at Dept. Biomedical Eng, Johns Hopkins Univ, Baltimore, MD 21205

[1]Examples of computational elements used by the nervous system to model inverse dynamics of a mechanical system were found by Shidara et al. (1993), where it was shown that the firing patterns of a set of Purkinje cells in the cerebellum could be reconstructed by an inverse dynamic representation of the eye.

### References

Brashers-Krug T, Shadmehr R, Todorov E (1995) Catastrophic interference in human motor learning. Adv Neural Inform Proc Syst, vol 7, in press.

Bruce RW (1933) Conditions of transfer of training. J Exp Psychol 16:343–361.

French, R. (1992) Semi-distributed Representations and Catastrophic Forgetting in Connectionist Networks, Connection Science 4:365-377.

Grafton ST et al. (1992) Functional anatomy of human procedural learning determined with regional cerebral blood flow and PET. J Neurosci 12:2542–2548.

Gandolfo F, Shadmehr R, Benda B, Bizzi E (1994) Adaptive behavior of the monkey motor system to virtual environments. Soc Neurosci Abs 20(2):1411.

Hogan N (1985) Impedance control: An approach for manipulation: Theory. J Dynam Sys Meas Cont 107:1–7.

Jezzard P et al. (1994) Practice makes perfect: A functional MRI study of long term motor cortex plasticity. 2nd Ann Soc. Magnetic Res., p. 330.

Kawashima R, Roland PE, O'Sullivan BT (1994) Fields in human motor areas involved in preparation for reaching, actual reaching, and visuomotor learning: A PET study. J Neurosci 14:3462–3474.

Lewis D, Shephard AH, Adams JA (1949) Evidences of associative interference in psychomotor performance. Science 110:271–273.

Melton AW, Irwin JM (1940) The influence of degree of interpolated learning on retroactive inhibition and the overt transfer of specific responses. Amer J Psychol 53:173–203.

Ross D (1974) Interference in discrete motor tasks: A test of the theory. PhD dissertation, Dept. Psychology, Univ. Michigan, Ann Arbor.

Schmidt RA (1988) Motor Control and Learning: A Behavioral Emphasis. Human Kinetics Books, Champaign IL, pp. 409–411.

Sears RR, Hovland CI (1941) Experiments on motor conflict. J Exp Psychol 28:280–286.

Shadmehr R, Mussa-Ivaldi FA (1994) Adaptive representation of dynamics during learning of a motor task. *J Neuroscience*, 14(5):3208–3224.

Shidara M, Kawano K, Gomi H, Kawato M (1993) Inverse dynamics model eye movement control by Purkinje cells in the cerebellum. Nature 365:50–52.

Siipola EM, Israel HE (1933) Habit interference as dependent upon stage of training. Amer J Psychol 45:205–227.

Squire LR (1986) Mechanisms of memory. Science 232:1612–1619.

Sutton RS (1986) Two problems with backpropagation and other steepest-descent learning procedures for networks. Proc 8th Cognitive Sci Soc, pp. 823–831.

Wolpert DM, Ghahramani Z, Jordan MI (1995) Are arm trajectories planned in kimenatic or dynamic coordinates? An adaptation study. Exp Brain Res, in press.
